# Inverse Dynamics
# of Speech Motor Control

Makoto Hirayama    Eric Vatikiotis-Bateson    Mitsuo Kawato*
ATR Human Information Processing Research Laboratories
2-2 Hikaridai, Seika-cho, Soraku-gun, Kyoto 619-02, Japan

## Abstract

Progress has been made in computational implementation of speech production based on physiological data. An inverse dynamics model of the speech articulator's musculo-skeletal system, which is the mapping from articulator trajectories to electromyographic (EMG) signals, was modeled using the acquired forward dynamics model and temporal (smoothness of EMG activation) and range constraints. This inverse dynamics model allows the use of a faster speech motor control scheme, which can be applied to phoneme-to-speech synthesis via musclo-skeletal system dynamics, or to future use in speech recognition. The forward acoustic model, which is the mapping from articulator trajectories to the acoustic parameters, was improved by adding velocity and voicing information inputs to distinguish acoustic parameter differences caused by changes in source characteristics.

## 1  INTRODUCTION

Modeling speech articulator dynamics is important not only for speech science, but also for speech processing. This is because many issues in speech phenomena, such as coarticulation or generation of aperiodic sources, are caused by temporal properties of speech articulator behavior due to musculo-skeletal system dynamics and constraints on neuro-motor command activation.

*Also, Laboratory of Parallel Distributed Processing, Research Institute for Electronic Science, Hokkaido University, Sapporo, Hokkaido 060, Japan

We have proposed using neural networks for a computational implementation of speech production based on physiological activities of speech articulator muscles. In previous works (Hirayama, Vatikiotis-Bateson, Kawato and Jordan 1992; Hirayama, Vatikiotis-Bateson, Honda, Koike and Kawato 1993), a neural network learned the forward dynamics, relating motor commands to muscles and the ensuing articulator behavior. From movement trajectories, the forward acoustic network generated the acoustic PARCOR parameters (Itakura and Saito, 1969) that were then used to synthesize the speech acoustics. A cascade neural network containing the forward dynamics model along with a suitable smoothness criterion was used to produce a continuous motor command from a sequence of discrete articulatory targets corresponding to the phoneme input string.

Along the same line, we have extended our model of speech motor control. In this paper, we focus on modeling the inverse dynamics of the musculo-skeletal system. Having an inverse dynamics model allows us to use a faster control scheme, which permits phoneme-to-speech synthesis via musculo-skeletal system dynamics, and ultimately may be useful in speech recognition. The final section of this paper reports improvements in the forward acoustic model, which were made by incorporating articulator velocity and voicing information to distinguish the acoustic parameter differences caused by changes in source characteristics.

## 2   INVERSE DYNAMICS MODELING OF MUSCULO-SKELETAL SYSTEM

From the viewpoint of control theory, an inverse dynamics model of a controlled object plays an essential role in feedforward control. That is, an accurate inverse dynamics model outputs an appropriate control sequence that realizes a given desired trajectory by using only feedforward control without any feedback information, so long as there is no perturbation from the environment. For speech articulators, the main control scheme cannot rely upon feedback control because of sensory feedback delays. Thus, we believe that the inverse dynamics model is essential for biological motor control of speech and for any efficient speech synthesis algorithm based on physiological data.

However, the speech articulator system is an excess-degrees-of-freedom system, thus the mapping from articulator trajectory (position, velocity, acceleration) to electromyographic (EMG) activity is one-to-many. That is, different EMG combinations exist for the same articulator trajectory (for example, co-contraction of agonist and antagonist muscle pairs). Consequently, we applied the forward modeling approach to learning an inverse model (Jordan and Rumelhart, 1992), i.e., constrained supervised learning, as shown in Figure 1. The inputs of the inverse

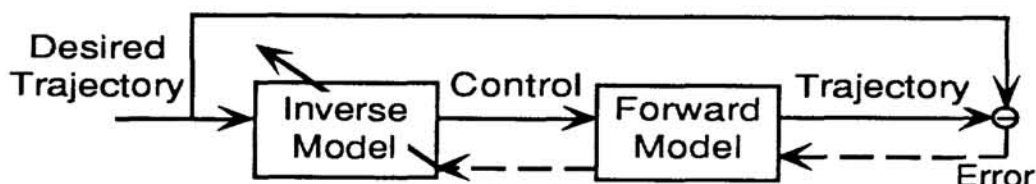

Figure 1: Inverse dynamics modeling using a forward dynamics model (Jordan and Rumelhart, 1992).

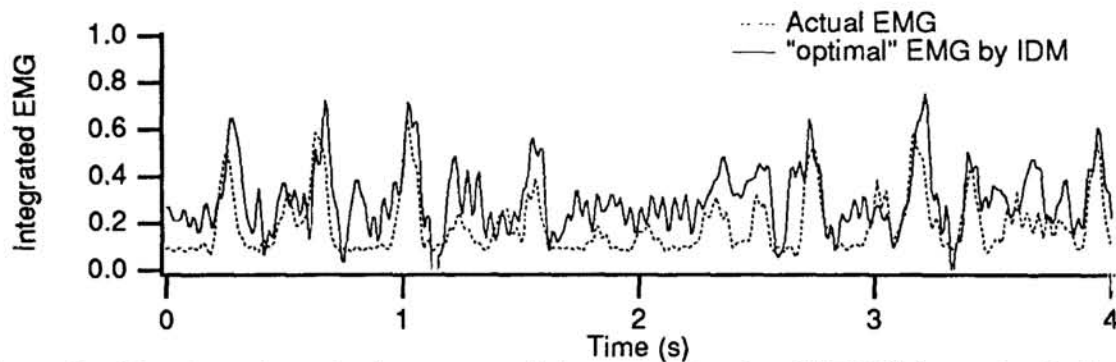

Figure 2: After learning, the inverse model output "optimal" EMG (anterior belly of the digastric) for jaw lowering is compared with actual EMG for the test trajectory.

dynamics model are articulator positions, velocities, and accelerations; the outputs are rectified, integrated, and filtered EMG for relevant muscles. The forward dynamics model previously reported (Hirayama et al., 1993) was used for determining the error signals of the inverse dynamics model.

To choose a realistic EMG pattern from among diverse possible solutions, we use both temporal and range constraints. The temporal constraint is related to the smoothness of EMG activation, i.e., minimizing EMG activation change (Uno, Suzuki, and Kawato, 1989). The minimum and maximum values of the range constraint were chosen using values obtained from the experimental data. Direct inverse modeling (Albus, 1975) was used to determine weights, which were then supplied as initial weights to the constrained supervised learning algorithm of Jordan and Rumelhart's (1992) inverse dynamics modeling method.

Figure 2 shows an example of the inverse dynamics model output after learning, when a real articulator trajectory, not included in the training set, was given as the input. Note that the network output cannot be exactly the same as the actual EMG, as the network chooses a unique "optimal" EMG from many possible EMG patterns that appear in the actual EMG for the trajectory.

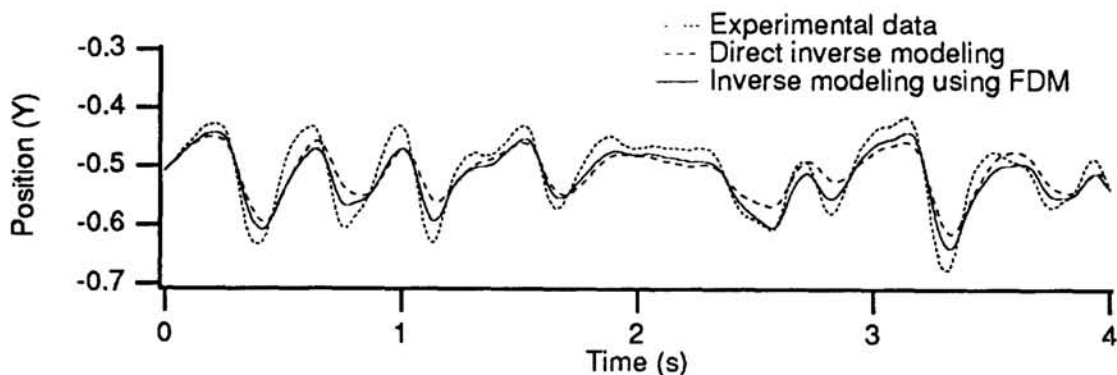

Figure 3: Trajectories generated by the forward dynamics network for the two methods of inverse dynamics modeling compared with the desired trajectory (experimental data).

Since the inverse dynamics model was obtained by learning, when the desired trajectory is given to the inverse dynamics model, an articulator trajectory can be generated with the forward dynamics network previously reported (Hirayama et al., 1993). Figure 3 compares trajectories generated by the forward dynamics network using EMG derived from the direct inverse dynamics method or the constrained supervised learning algorithm (which uses the forward dynamics model to determine the inverse dynamics model's "optimal" EMG). The latter method yielded a 30.0 % average reduction in acceleration prediction error over the direct method, thereby bringing the model output trajectory closer to the experimental data.

## 3    TRAJECTORY FORMATION USING FORWARD AND INVERSE RELAXATION MODEL

Previously, to generate a trajectory from discrete phoneme-specific via-points, we used a cascade neural network (c.f., Hirayama et al., 1992). The inverse dynamics model allows us to use an alternative network proposed by Wada and Kawato (1993) (Figure 4). The network uses both the forward and inverse models of the controlled object, and updates a given initial rough trajectory passing through the via-points according to the dynamics of the controlled object and a smoothness constraint on the control input. The computation time of the network is much shorter than that of the cascade neural network (Wada and Kawato, 1993).

Figure 5 shows a forward dynamics model output trajectory driven by the model-generated motor control signals. Unlike Wada and Kawato's original model (1993) in which generated trajectories always pass through via-points, our trajectories were generated from smoothed motor control signals (i.e., after applying the smoothness constraint) and, consequently, do not pass through the exact via-points. In this paper, a typical value for each phoneme from experimental data was chosen as the target via-point and was given in Cartesian coordinates relative to the maxillary incisor. Although further investigation is needed to refine the phoneme-specific target specifications (e.g. lip aperture targets), reasonable coarticulated trajectories were obtained from series of discrete via-point targets (Figure 5). For engineering applications such as text-to-speech synthesizers using articulatory synthesis, this kind of technique is necessary because realistic coarticulated trajectories must serve as input to the articulatory synthesizer.

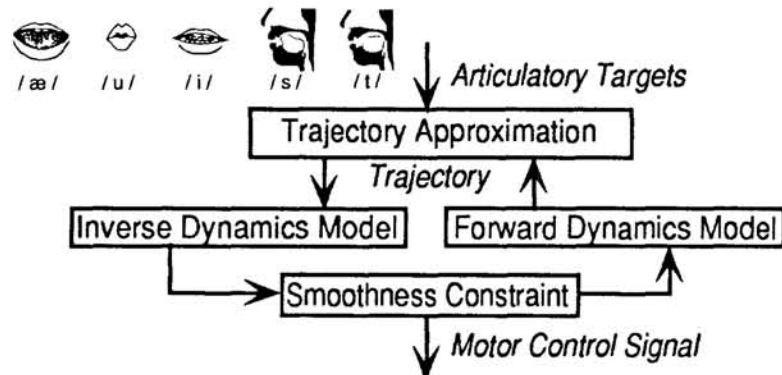

Figure 4: Speech trajectory formation scheme modified from the forward and inverse relaxation neural network model (Wada and Kawato, 1993).

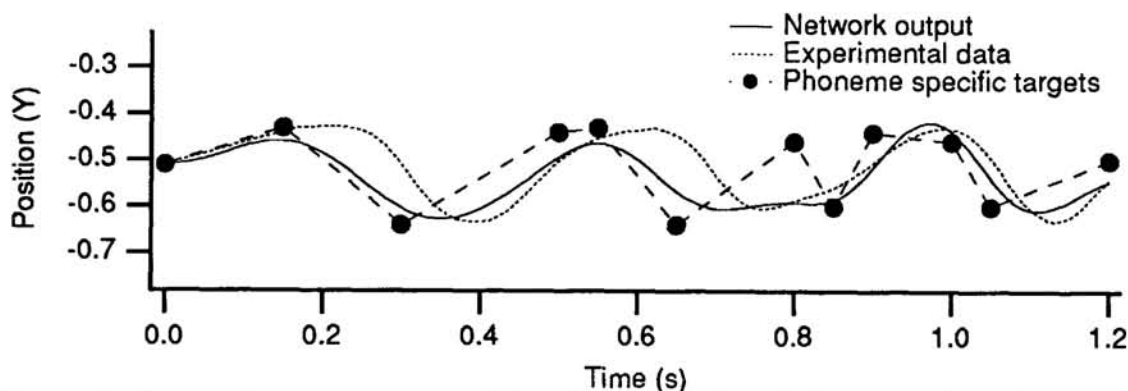

Figure 5: Jaw trajectory generated by the forward and inverse relaxation model. The output of the forward dynamics model is used for this plot.

A further advantage of this network is that it can be used to predict phoneme-specific via-points from the realized trajectory (Wada, Koike, Vatikiotis-Bateson and Kawato, 1993). This capability will allow us to use our forward and inverse dynamics models for speech recognition in future, through acoustic to articulatory mapping (Shirai and Kobayashi, 1991; Papcun, Hochberg, Thomas, Laroche, Zacks and Levy, 1992) and the articulatory to phoneme specific via-points mapping discussed above. Because trajectories may be recovered from a small set of phoneme-specific via-points, this approach should be readily applicable to problems of speech data compression.

## 4   DYNAMIC MODELING OF FORWARD ACOUSTICS

The second area of progress is the improvement in the forward acoustic network. Previously (Hirayama et al., 1993), we demonstrated that acoustic signals can be obtained using a neural network that learns the mapping between articulator positions and acoustic PARCOR coefficients (Itakura and Saito, 1969; See also, Markel and Gray, 1976).

However, this modeling was effective only for vowels and a limited number of consonants because the architecture of the model was basically the same as that of static articulatory synthesizers (e.g. Mermelstein, 1973). For natural speech, aperiodic sources for plosive and sibilant consonants result in multiple sets of acoustic parameters for the same articulator configuration (i.e., the mapping is one-to-many); hence, learning did not fully converge. One approach to solving this problem is to make source modeling completely separate from the vocal tract area modeling. However, for synthesis of natural sentences, the vocal tract transfer function model requires another model for the non-glottal sources associated with consonant production. Since these sources are located at various points along the vocal tract, their interaction is extremely complex.

Our approach to solving this one-to-many mapping is to have the neural network learn the acoustic parameters along with the sound source characteristic specific to each phoneme. Thus, we put articulator positions with their velocities and voiced/voiceless information (e.g., Markel and Gray, 1976) into the input (Figure 6) because the sound source characteristics are made not only by the articulator posi-

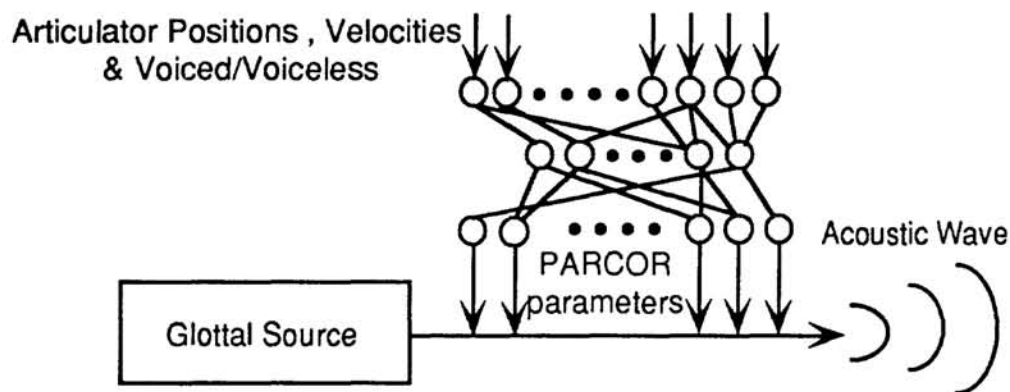

Figure 6: Improved forward acoustic network. Inputs to the network are articulator positions and velocities and voiced/voiceless information.

tion but also by the dynamic movement of articulators.

For simulations, horizontal and vertical motions of jaw, upper and lower lips, and tongue tip and blade were used for the inputs and 12 dimensional PARCOR parameters were used for the outputs of the network. Figure 7(a) shows position-velocity-voiced/voiceless network output compared with position-only network and experimentally obtained PARCOR parameters for a natural test sentence. Only the first two coefficients are shown. The first part of the test sentence, "Sam sat on top of the potato cooker and waited for Tommy to cut up a bag of tiny tomatoes and pop the beat tips into the pot," is shown in this plot. Figure 7(b)(c) show a part of the synthesized speech driven by fundamental frequency pulses for voiced sounds and random noises for voiceless sounds.

By using velocity and voiced/voiceless inputs, the performance was improved for natural utterances which include many vowels and consonants. The average values of the LPC-cepstrum distance measure between original and synthesized, were 5.17 (dB) for the position-only network and 4.18 (dB) for the position-velocity-voiced/voiceless network. When listening to the output, the sentence can be understood, and almost all vowels and many of the consonants can be classified. The overall clarity and the classification of some consonants is about as difficult as experienced in noisy international telephone calls.

Although there are other potential means to achieve further improvement (e.g. adding more tongue channels, using more balanced training patterns, incorporating nasality information, implementation of better glottal and non-glottal sources), the network synthesizes quite smooth and reasonable acoustic signals by incorporating aspects of the articulator dynamics.

## 5    CONCLUSION

We are modeling the information transfer from phoneme-specific articulatory targets to acoustic wave via the musculo-skeletal system, using a series of neural networks. Electromyographic (EMG) signals are used as the reflection of motor control commands. In this paper, we have focused on the inverse dynamics modeling of the

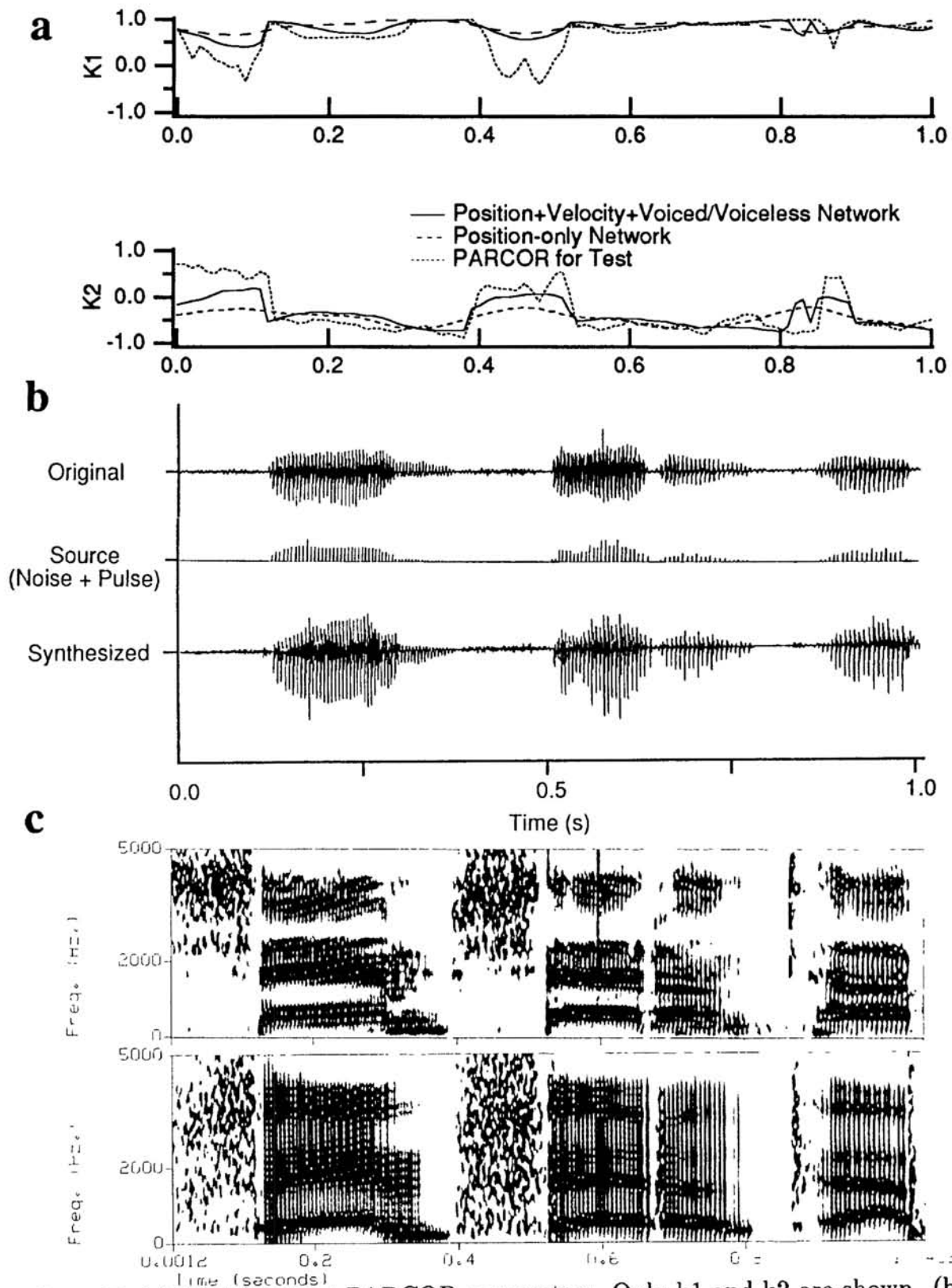

Figure 7: (a) Model output PARCOR parameters. Only k1 and k2 are shown. (b) Original, source model, and synthesized acoustic signals. (c) Wideband spectrogram for the original and synthesized speech. Utterance shown is "Sam sat on top" from a test sentence.

musculo-skeletal system, its control for the transform from discrete linguistic information to continuous motor control signals, and articulatory speech synthesis using the articulator dynamics. We believe that modeling the dynamics of articulatory motions is a key issue both for elucidating mechanisms of speech motor control and for synthesis of *natural* utterances.

## Acknowledgements

We thank Yoh'ichi Toh'kura for continuous encouragement. Further support was provided by HFSP grants to M. Kawato.

## References

Albus, J. S. (1975) A new approach to manipulator control: The cerebellar model articulation controller (CMAC). *Transactions of the ASME Journal of Dynamic System, Measurement, and Control*, 220-227.

Hirayama, M., E. Vatikiotis-Bateson, M. Kawato, and M. I. Jordan (1992) Forward dynamics modeling of speech motor control using physiological data. In Moody, J. E., Hanson, S. J., and Lippmann, R. P. (eds.) *Advances in Neural Information Processing Systems 4*. San Mateo, CA: Morgan Kaufmann Publishers, 191-198.

Hirayama, M., E. Vatikiotis-Bateson, K. Honda, Y. Koike, and M. Kawato (1993) Physiologically based speech synthesis. In Giles, C. L., Hanson, S. J., and Cowan, J. D. (eds.) *Advances in Neural Information Processing Systems 5*. San Mateo, CA: Morgan Kaufmann Publishers, 658-665.

Itakura, F. and S. Saito (1969) Speech analysis and synthesis by partial correlation parameters. *Proceeding of Japan Acoustic Society*, **2-2-6** (In Japanese).

Jordan, M. I. and D. E. Rumelhart (1992) Forward models: Supervised learning with a distal teacher. *Cognitive Science*, **16**, 307-354.

Mermelstein, P. (1973) Articulatory model for the study of speech production. *Journal of Acoustical Society of America*, **53**, 1070-1082.

Papcun, J., J. Hochberg, T. R. Thomas, T. Laroche, J. Zacks, and S. Levy (1992) Inferring articulation and recognizing gestures from acoustics with a neural network trained on x-ray microbeam data. *Journal of Acoustical Society of America*, **92 (2) Pt. 1**.

Shirai, K. and T. Kobayashi (1991) Estimation of articulatory motion using neural networks. *Journal of Phonetics*, **19**, 379-385.

Uno, Y., R. Suzuki, and M. Kawato (1989) The minimum muscle tension change model which reproduces arm movement trajectories. *Proceeding of the 4th Symposium on Biological and Physiological Engineering*, 299-302 (In Japanese).

Wada, Y. and M. Kawato (1993) A neural network model for arm trajectory formation of using forward and inverse dynamics models. *Neural Networks*, **6**, 919-932.

Wada, Y., Y. Koike, E. Vatikiotis-Bateson, and M. Kawato (1993) Movement Pattern Recognition Based on the Minimization Principle. *Technical Report of IEICE*, **NC93-23**, 85-92 (In Japanese).